# Discovering Discrete Distributed Representations with Iterative Competitive Learning

**Michael C. Mozer**
Department of Computer Science
and Institute of Cognitive Science
University of Colorado
Boulder, CO 80309-0430

## Abstract

Competitive learning is an unsupervised algorithm that classifies input patterns into mutually exclusive clusters. In a neural net framework, each cluster is represented by a processing unit that competes with others in a winner-take-all pool for an input pattern. I present a simple extension to the algorithm that allows it to construct discrete, *distributed* representations. Discrete representations are useful because they are relatively easy to analyze and their information content can readily be measured. Distributed representations are useful because they explicitly encode similarity. The basic idea is to apply competitive learning iteratively to an input pattern, and after each stage to subtract from the input pattern the component that was captured in the representation at that stage. This component is simply the weight vector of the winning unit of the competitive pool. The subtraction procedure forces competitive pools at different stages to encode different aspects of the input. The algorithm is essentially the same as a traditional data compression technique known as multistep vector quantization, although the neural net perspective suggests potentially powerful extensions to that approach.

## 1 INTRODUCTION

Competitive learning (Grossberg, 1976; Kohonen, 1982; Rumelhart & Zipser, 1985; von der Malsburg, 1973) is an unsupervised algorithm that classifies input patterns into mutually exclusive clusters. In a neural net framework, each cluster is represented by a processing unit that competes with others in a winner-take-all pool for each input pattern. Competitive learning thus constructs a local representation in which a single unit is activated in response to an input. I present a simple extension to the algorithm that allows it to construct discrete, *distributed* representations. Discrete representations are useful because they are relatively easy to analyze and their information content can readily be measured. Distributed representations are useful because they explicitly encode similarity. I begin by describing the standard competitive learning algorithm.

## 2  COMPETITIVE LEARNING

Consider a two layer network with $\alpha$ input units and $\beta$ competitive units. Each competitive unit represents a different classification of the input. The competitive units are activated by the input units and are connected in a winner-take-all pool such that a single competitive unit becomes active. Formally,

$$y_i = \begin{cases} 1 & \text{if } |\mathbf{w}_i-\mathbf{x}| \le |\mathbf{w}_j-\mathbf{x}| \text{ for all } j \\ 0 & \text{otherwise,} \end{cases}$$

where $y_i$ is the activity of competitive unit $i$, $\mathbf{x}$ is the input activity vector, $\mathbf{w}_i$ is the vector of connection strengths from the input units to competitive unit $i$, and $|\cdot|$ denotes the L2 vector norm. The conventional weight update rule is:

$$\Delta\mathbf{w}_i = \varepsilon y_i(\mathbf{x}-\mathbf{w}_i),$$

where $\varepsilon$ is the step size. This algorithm moves each weight vector toward the center of a cluster of input patterns.

The algorithm attempts to develop the best possible representation of the input with only $\beta$ discrete alternatives. This representation is simply the weight vector of the winning competitive unit, $\mathbf{w}_{winner}$. What does it mean to develop the *best* representation? Following Durbin (1990), competitive learning can be viewed as performing gradient descent in the error measure

$$E = -\sum_{p=1}^{\#patterns} \ln\sum_{i=1}^{\beta} e^{-|\mathbf{w}_i-\mathbf{x}(p)|^2/T} \tag{1}$$

as $T\to0$, where $p$ is an index over patterns. $T$ is a parameter in a soft competitive learning model (Bridle, 1989; Rumelhart, in press) which specifies the degree of competition; the winner-take-all version of competitive learning is obtained at the limit of $T = 0$.

## 3  EXTENDING COMPETITIVE LEARNING

Competitive learning constructs a *local* representation of the input. How might competitive learning be extended to construct *distributed* representations? One idea is to have several independent competitive pools, each of which may form its own partition of the input space. This often fails because all pools will discover the *same* partitioning if this partitioning is unequivocally better than others. Thus, we must force different pools to encode different components of the input.

In the one-pool competitive learning network, the component of the input not encoded is simply

$$\mathbf{x}' = \mathbf{x} - \mathbf{w}_{winner}.$$

If competitive learning is reapplied with $\mathbf{x}'$ instead of $\mathbf{x}$, the algorithm is guaranteed to extract information not captured by the first pool of competitive units because this information has been subtracted out. This procedure can be invoked iteratively to capture different aspects of the input in an arbitrary number of competitive pools, hence the name *iterative competitive learning* or *ICL*. The same idea is at the heart of Sanger's (1989) and Hrycej's (1989) algorithms for performing principal components analysis. Whereas these algorithms discover continuous-valued feature dimensions, ICL is concerned with

the discovery of discrete-valued features. Of course, the continuous features can be quantized to form discrete features, an idea that both Sanger and Hrycej explore, but there is a cost to this, as I elaborate later.

To formalize the ICL model, consider a network composed of an arbitrary number of *stages* (Figure 1). Each stage, $s$, consists of $\alpha$ input units and $\beta^{(s)}$ competitive units. Both the input and competitive units at a given stage feed activity to the input units at the next higher stage. The activity of the input units at stage 1, $\mathbf{x}^{(1)}$, is given by the external input. At subsequent stages, $s$,

$$\mathbf{x}^{(s)} = \mathbf{x}^{(s-1)} - \left[\mathbf{W}^{(s-1)}\right]^{\mathbf{T}} \mathbf{y}^{(s-1)}$$

where $\mathbf{W}$ and $\mathbf{y}$ are as before with an additional index for the stage number.

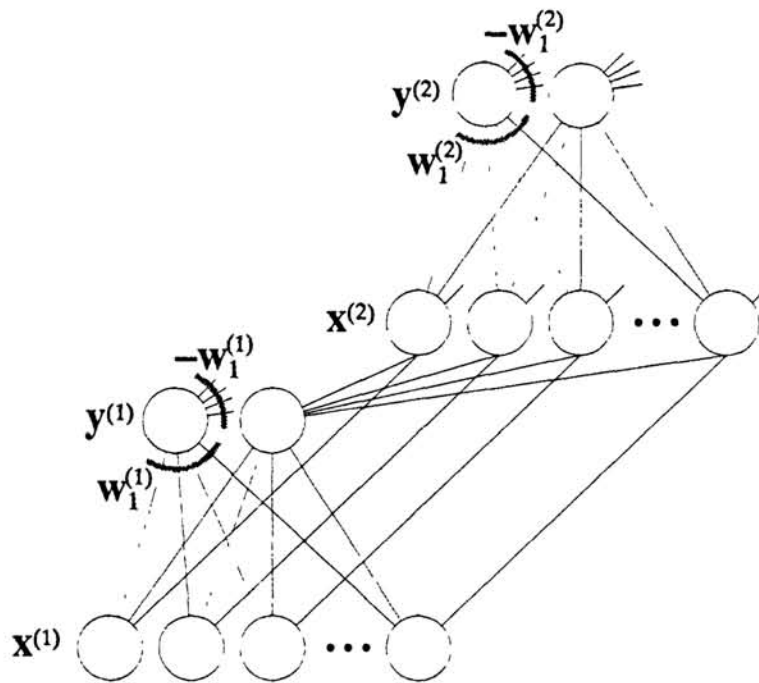

Figure 1:  The Iterative Competitive Learning Model

To reconstruct the original input pattern from the activities of the competitive units, the components captured by the winning unit at each stage are simply summed together:

$$\hat{\mathbf{x}} = \sum_{s} \left[\mathbf{W}^{(s)}\right]^{\mathbf{T}} \mathbf{y}^{(s)}. \tag{2}$$

A variant of ICL has been independently proposed by Ambros-Ingerson, Granger, and Lynch (1990).[1] Their algorithm, inspired by a neurobiological model, is the same as ICL except for the competitive unit activation rule which uses an inner product instead of distance measure:

$$y_i = \begin{cases} 1 & \text{if } x^T w_i \geq 0 \text{ and } x^T w_i \geq x^T w_j \text{ for all } j \\ 0 & \text{otherwise.} \end{cases}$$

The problem with this rule is that it is difficult to interpret what exactly the network is computing, e.g., what aspect of the input is captured by the winning unit, whether the input can be reconstructed from the resulting activity pattern, and what information is discarded. The ICL activation rule, in combination with the learning rule, has a clear computational justification by virtue of the underlying objective measure (Equation 1) that is being optimized.

It also turns out, much to my dismay, that ICL is virtually identical to a conventional technique in data compression known as multistep vector quantization (Gray, 1984). More on this later.

## 4   A SIMPLE EXAMPLE

Consider a set of four input patterns forming a rectangle in 2D space, located at $(-1,-.5)$, $(-1,.5)$, $(1,-.5)$, and $(1,.5)$, and an ICL network with two stages each containing two competitive units. The first stage discovers the primary dimension of variation — along the x-axis. That is, the units develop weight vectors $(-1,0)$ and $(1,0)$. Removing this component from the input, the four points become $(0,-.5)$, $(0,.5)$, $(0,-.5)$, $(0,.5)$. Thus, the two points on the left side of the rectangle are collapsed together with the two points on the right side. The second stage of the network then discovers the secondary dimension of variation — along the y-axis.

The response of the ICL network to each input pattern can be summarized by the set of competitive units, one per stage, that are activated. If the two units at each stage are numbered 0 and 1, four response patterns will be generated: {0,0}, {0,1}, {1,0}, {1,1}. Thus, ICL has discovered a two-bit code to represent the four inputs. The result will be the same if instead of just four inputs, the input environment consists of four *clusters* of points centered on the corners of the rectangle. In this case, the two-bit code will not describe each input uniquely, but it will distinguish the clusters.

## 5   IMAGE COMPRESSION

Because ICL discovers compact and discrete codes, the algorithm should be useful for data and image compression. In such problems, a set of raw data must be transformed into a compact representation which can then be used to reconstruct the original data. ICL performs such a transformation, with the resulting code consisting of the competitive unit response pattern. The reconstruction is achieved by Equation 2.

I experimented with a 600×460 pixel image having 8 bits of gray level information per pixel. ICL was trained on random 8×8 patches of the image for a total of 125,000 training trials. The network had 64 input units and 80 stages, each with two competitive units. The initial weights were random, selected from a Normal distribution with mean zero and standard deviation .0001. A fixed $\varepsilon$ of .01 was used. Figure 2 shows incoming connection strengths to the competitive units in the first nine stages. The connection strengths are depicted as an 8×8 grid of cells whose shading indicates the weight from the corresponding position in the image patch to the competitive unit.

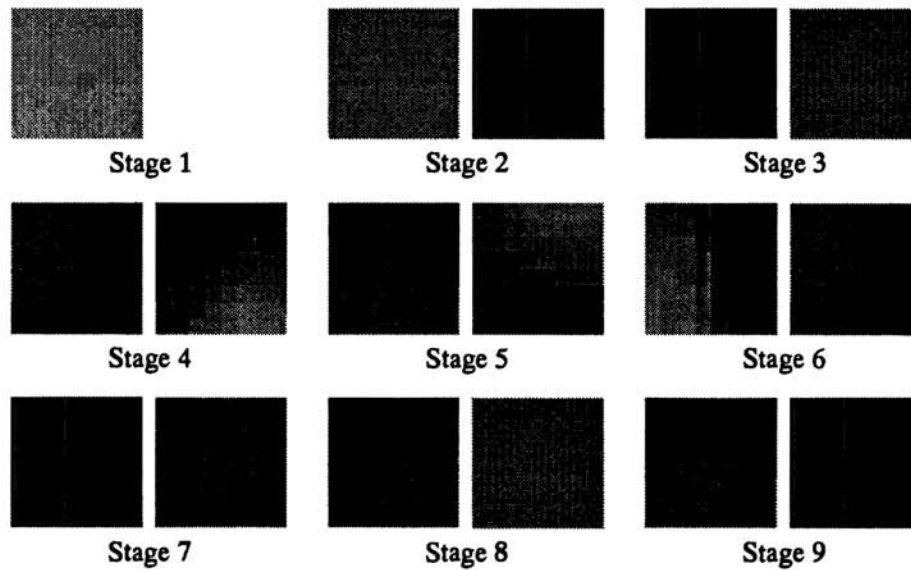

Figure 2: Input-to-Competitive Unit Connection Strengths at Stages 1-9

Following training, the image is compressed by dividing the image into nonoverlapping 8×8 patches, presenting each in turn to ICL, obtaining the compressed code, and then reconstructing the patch from the code. With an $s$ stage network and two units per stage, the compressed code contains $s$ bits. Thus, the number of bits per pixel in the compressed code is $s/(8\times8)$. To obtain different levels of compression, the number of stages in ICL can be varied. Fortunately, this does not require retraining ICL because the features detected at each stage do not depend on the number of stages; the earlier stages capture the most significant variation in the input. Thus, if the network is trained with 80 stages, one can use just the first 32 to compress the image, achieving a .5 bit per pixel encoding.

The image used to train ICL was originally used in a neural net image compression study by Cottrell, Munro, and Zipser (1989). Their compression scheme used a three-layer back propagation autoencoder to map an image patch back to itself through a hidden layer. The hidden layer, with fewer units than the input layer, served as the encoding. Because hidden unit activities are continuous valued, it was necessary to quantize the activities. Using a standard measure of performance, the signal-to-noise ratio (the logarithm of the average energy relative to the average reconstruction error), ICL outperforms Cottrell et al.'s network (Table 1).

This result is not surprising. In the data compression literature, vector quantization approaches — similar to ICL — usually work better than transformation-based approaches — e.g., Cottrell et al. (1989), Sanger (1989). The reason is that transformation-based approaches do not take quantization into account in the development of the code. That is, in transformation-based approaches, the training procedure, which discovers the code, and the quantization step, which turns this code into a form that can be used for digital

data transmission or storage, are two distinct processes. In Cottrell et al.'s network, a hidden unit encoding is learned without considering the demands of quantization. There is no assurance that the quantized code will retain the information in the signal. In contrast, ICL takes quantization into account *during* training.

**Table 1:** Signal-to-Noise Ratio for Different Compression Levels

| compression | Cottrell et al. | ICL |
|---|---|---|
| 1.25 bits/pixel | 2.324 | 2.366 |
| 1 bit/pixel | 2.170 | 2.270 |
| .75 bits/pixel | 1.746 | 2.146 |
| .5 bits/pixel | not available | 1.975 |

# 6   COMPARISON TO VECTOR QUANTIZATION APPROACHES

As I mentioned previously, ICL is essentially a neural net reformulation of a conventional data compression scheme called multistep vector quantization. However, adopting a neural net perspective suggests several promising variants of the approach. These variants result from viewing the encoding task as an optimization problem (i.e., finding weights that minimize Equation 1). I mention three variants, the first two of which are methods for finding the solution more efficiently and consistently, the final one is a powerful extension to algorithm that I believe has not yet been studied in the vector quantization literature.

## 6.1   AVOIDING LOCAL OPTIMA

As Rumelhart and Zipser (1985) and others have noted, competitive learning experiences a serious problem from locally optimal solutions in which one competitive unit captures most or all of the input patterns while others capture none. To eliminate such situations, I have introduced a secondary error term whose purpose is to force the competitive units to win equally often:

$$E_{sec} = \sum_{i=1}^{\beta} (\frac{1}{\beta} - \bar{y}_i)^2 ,$$

where $\bar{y}_i$ is the mean activity of competitive unit $i$ over trials. Based on the soft competitive learning model with $T > 0$, this yields the weight update rule

$$\Delta w_i = \gamma (x - w_i)(1 - \beta \bar{y}_i) ,$$

where $\gamma$ is the step size. Because this constraint should not be part of the ultimate solution, $\gamma$ must gradually be reduced to zero. In the image compression simulation, $\gamma$ was set to .005 initially and was decreased by .0001 every 100 training trials. This is a more principled solution to the local optimum problem than the "leaky learning" idea suggested by Rumelhart and Zipser. It can also be seen as an alternative or supplement to the schemes proposed for selecting the initial code (weights) in the vector quantization literature.

## 6.2 CONSTRAINTS ON THE WEIGHTS

I have explored a further idea to increase the likelihood of converging on a good solution and to achieve more rapid convergence. The idea is based on two facts. First, in an optimal solution, the weight vector of a competitive unit should be the mean of the inputs captured by that unit. This gives rise to the second observation: beyond stage 1, the mean input, $\bar{x}^{(s)}$, should be zero.

If the competitive pools contain two units, these facts lead to a strong constraint on the weights:

$$0 = \bar{x}^{(s)}$$

$$= \frac{\sum_{p \in PART_1} x^{(s)}(p) + \sum_{p \in PART_2} x^{(s)}(p)}{n_1 + n_2}$$

$$= \frac{n_1 w_1^{(s)} + n_2 w_2^{(s)}}{n_1 + n_2},$$

where $x^{(s)}(p)$ is the input vector in stage $s$ for pattern $p$, $PART_1$ and $PART_2$ are the two clusters of input patterns partitioned by the competitive units at stage $s-1$, and $n_1$ and $n_2$ are the number of elements in each cluster.

The consequence is that, in an optimal solution,

$$w_1 = -\frac{n_2}{n_1} w_2 .$$

(This property is observed in Figure 2.) Constraining the weights in this manner, and performing gradient descent in the ratio $n_2/n_1$, as well as in the weight parameters themselves, the quality of the solution and the convergence rate are dramatically improved.

## 6.3 GENERALIZING THE TRANSFORMATION BETWEEN STAGES

At each stage $s$, the winning competitive unit specifies a transformation of $x^{(s)}$ to obtain $x^{(s+1)}$. In ICL, this transformation is simply a translation. There is no reason why this could not be generalized to include rotation and dilation as well, i.e.,

$$x^{(s+1)} = T_{winner}^{(s)} x^{(s)},$$

where $T_{winner}$ is a transformation matrix that includes the translation specified by $w_{winner}$. (For this notation to be formally correct, $x$ must be augmented by an element having constant value 1 to allow for translations.) The rotation and dilation parameters can be learned via gradient descent search in the error measure given in Equation 1. Reconstruction involves inverting the sequence of transformations:

$$\hat{x} = \left[ T_{winner}^{(1)} \right]^{-1} \cdots \left[ T_{winner}^{(s)} \right]^{-1} (0\ 0\ 0 \cdots 1)^T .$$

A simple example of a situation in which this generalized transformation can be useful is depicted in Figure 3. After subtracting out the component detected at stage 1, the two clusters may be rotated into alignment, allowing the second stage to capture the remain-

ing variation in the input. Whether or not this extension proves useful has yet to be tested. However, the connectivity patterns in Figure 2 certainly suggest that factoring out variations in orientation might permit an even more compact representation of the input data.

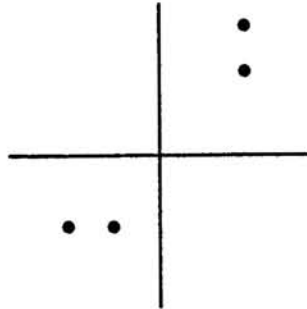

Figure 3:  A Sample Input Space With Four Data Points

## Acknowledgements

This research was supported by NSF grant IRI-9058450 and grant 90-21 from the James S. McDonnell Foundation.  My thanks to Paul Smolensky for helpful comments on this work and to Gary Cottrell for providing the image data and associated software.

## Footnotes

[1] I thank Todd Leen and Steve Rehfuss for bringing this work to my attention.

## References

Ambros-Ingerson, J., Granger, G., & Lynch, G. (1990).  Simulation of paleocortex performs hierarchical clustering. *Science, 247,* 1344-1348.

Bridle, J. (1990).  Training stochastic model recognition algorithms as networks can lead to maximum mutual information estimation of parameters. In D. S. Touretzky (Ed.), *Advances in neural information processing systems 2* (pp. 211–217).  San Mateo, CA:  Morgan Kaufmann.

Cottrell, G. W., Munro, P., & Zipser, D. (1989).  Image compression by back propagation:  An example of extensional programming.  In N. Sharkey (Ed.), *Models of cognition: A review of cognitive science* (pp. 208–240).  Norwood, NJ:  Ablex.

Durbin, R. (April, 1990).  Principled competitive learning in both unsupervised and supervised networks.  Poster presented at the conference on Neural Networks for Computing, Snowbird, Utah.

Gray, R. M. (1984).  Vector quantization.  *IEEE ASSP Magazine,* 4–29.

Grossberg, S. (1976).  Adaptive pattern classification and universal recoding.  I:  Parallel development and coding of neural feature detectors.  *Biological Cybernetics, 23,* 121–134.

Hrycej, T. (1989).  Unsupervised learning by backward inhibition.  *Proceedings of the Eleventh International Joint Conference on Artificial Intelligence* (pp. 170-175).  Los Altos, CA:  Morgan Kaufmann.

Kohonen, T. (1982).  Clustering, taxonomy, and topological maps of patterns.  In M. Lang (Ed.), *Proceedings of the Sixth International Conference on Pattern Recognition* (pp. 114–125).  Silver Spring, MD:  IEEE Computer Society Press.

Rumelhart, D. E. (in press).  Connectionist processing and learning as statistical inference.  In Y. Chauvin & D. E. Rumelhart (Eds.), *Backpropagation: Theory, architectures, and applications.* Hillsdale, NJ:  Erlbaum.

Rumelhart, D. E., & Zipser, D. (1985).  Feature discovery by competitive learning.  *Cognitive Science, 9,* 75–112.

Sanger, T. D. (1989).  Optimal unsupervised learning in a single-layer linear feedforward neural network.  *Neural Networks, 2,* 459–473.

von der Malsburg, C. (1973).  Self-organization of orientation sensitive cells in the striate cortex.  *Kybernetik, 14,* 85–100.